# Conditional Neural Fields

**Jian Peng**
Toyota Technological Institute at Chicago
6045 S. Kenwood Ave.
Chicago, IL 60637
jpengwhu@gmail.com

**Liefeng Bo**
Toyota Technological Institute at Chicago
6045 S. Kenwood Ave.
Chicago, IL 60637
liefengbo@gmail.com

**Jinbo Xu**
Toyota Technological Institute at Chicago
6045 S. Kenwood Ave.
Chicago, IL 60637
jinboxu@gmail.com

## Abstract

Conditional random fields (CRF) are widely used for sequence labeling such as natural language processing and biological sequence analysis. Most CRF models use a linear potential function to represent the relationship between input features and output. However, in many real-world applications such as protein structure prediction and handwriting recognition, the relationship between input features and output is highly complex and nonlinear, which cannot be accurately modeled by a linear function. To model the nonlinear relationship between input and output we propose a new conditional probabilistic graphical model, Conditional Neural Fields (CNF), for sequence labeling. CNF extends CRF by adding one (or possibly more) middle layer between input and output. The middle layer consists of a number of gate functions, each acting as a local neuron or feature extractor to capture the nonlinear relationship between input and output. Therefore, conceptually CNF is much more expressive than CRF. Experiments on two widely-used benchmarks indicate that CNF performs significantly better than a number of popular methods. In particular, CNF is the best among approximately 10 machine learning methods for protein secondary structure prediction and also among a few of the best methods for handwriting recognition.

## 1 Introduction

Sequence labeling is a ubiquitous problem arising in many areas, including natural language processing [1], bioinformatics [2, 3, 4] and computer vision [5]. Given an input/observation sequence, the goal of sequence labeling is to infer the state sequence (also called output sequence), where a state may be some type of labeling or segmentation. For example, in protein secondary structure prediction, the observation is a protein sequence consisting of a collection of residues. The output is a sequence of secondary structure types. Hidden Markov model (HMM) [6] is one of the popular methods for sequence labeling. HMM is a generative learning model since it generates output from a joint distribution between input and output. In the past decade, several discriminative learning models such as conditional random fields (CRF) have emerged as the mainstream methods for sequence labeling. Conditional random fields, introduced by Lafferty [7], is an undirected graphical model. It defines the conditional probability of the output given the input. CRF is also a special case of the log-linear model since its potential function is defined as a linear combination of features. Another approach for sequence labeling is max margin structured learning such as max margin Markov

networks (MMMN) [8] and SVM-struct [9]. These models generalize the large margin and kernel methods to structured learning.

In this work, we present a new probabilistic graphical model, called conditional neural fields (CNF), for sequence labeling. CNF combines the advantages of both CRF and neural networks. First, CNF preserves the globally consistent prediction, i.e. exploiting the structural correlation between outputs, and the strength of CRF as a rigorous probabilistic model. Within the probabilistic framework, posterior probability can be derived to evaluate confidence on predictions. This property is particularly valuable in applications that require multiple cascade predictors. Second, CNF automatically learns an implicit nonlinear representation of features and thus, can capture more complicated relationship between input and output. Finally, CNF is much more efficient than kernel-based methods such as MMMN and SVM-struct. The learning and inference procedures in CNF adopt efficient dynamic programming algorithm, which makes CNF applicable to large scale tasks.

## 2    Conditional Random Fields

Assume the input and output sequences are $X$ and $Y$, respectively. Meanwhile, $Y = \{y_1, y_2, ..., y_N\} \in \Sigma^N$ where $\Sigma$ is the alphabet of all possible output states and $|\Sigma| = M$.

CRF uses two types of features given a pair of input and output sequences. The first type of features describes the dependency between the neighboring output labels.

$$f_{y,y'}(Y, X, t) = \delta[y_t = y]\delta[y_{t-1} = y'] \tag{1}$$

where $\delta[y_t = y]$ is a indicator function. It is equal to 1 if and only if the state at position $t$ is $y$.

The second type of features describes the dependency between the label at one position and the observations around this position.

$$f_y(Y, X, t) = \mathbf{f}(X, t)\delta[y_t = y] \tag{2}$$

where $\mathbf{f}(X,t)$ is the local observation or feature vector at position $t$.

In a linear chain CRF model [7], the conditional probability of the output sequence $Y$ given the input sequence $X$ is the normalized product of the exponentials of potential functions on all edges and vertices in the chain.

$$P(Y|X) = \frac{1}{Z(X)}\mathbf{exp}(\sum_{t=1}^{N}(\psi(Y, X, t) + \phi(Y, X, t))) \tag{3}$$

where

$$\phi(Y, X, t) = \sum_y w_y^T f_y(Y, X, t) \tag{4}$$

is the potential function defined on vertex at the $t^{th}$ position, which measures the compatibility between the local observations around the $t^{th}$ position and the output label $y_t$; and

$$\psi(Y, X, t) = \sum_{y,y'} u_{y,y'} f_{y,y'}(Y, X, t) \tag{5}$$

is the potential function defined on an edge connecting two labels $y_t$ and $y_{t+1}$. This potential measures the compatibility between two neighbor output labels.

Although CRF is a very powerful model for sequence labeling, CRF does not work very well on the tasks in which the input features and output labels have complex relationship. For example, in computer vision or bioinformatics, many problems require the modeling of complex/nonlinear relationship between input and output [10, 11]. To model complex/nonlinear relationship between input and output, CRF has to explicitly enumerate all possible combinations of input features and output labels. Nevertheless, even assisted with domain knowledge, it is not always possible for CRF to capture all the important nonlinear relationship by explicit enumeration.

## 3    Conditional Neural Fields

Here we propose a new probabilistic graphical model, conditional neural fields (CNF), for sequence labeling. Figure 1 shows the structural difference between CNF and CRF. CNF not only

can parametrize the conditional probability in the log-linear like formulation, but also is able to implicitly model complex/nonlinear relationship between input features and output labels. In a linear chain CNF, the edge potential function is similar to that of a linear chain CRF. That is, the edge function describes only the interdependency between the neighbor output labels. However, the potential function of CNF at each vertex is different from that of CRF. The function is defined as follows.

$$\phi[Y, X, t] = \sum_y \sum_{g=1}^{K} w_{y,g} h(\theta_g^T \mathbf{f}(X, t)) \delta[y_t = y] \tag{6}$$

where $h$ is a gate function. In this work, we use the logistic function as the gate function. The major difference between CRF and CNF is the definition of the potential function at each vertex. In CRF, the local potential function (see Equation (4)) is defined as a linear combination of features. In CNF, there is an extra hidden layer between the input and output, which consists of $K$ gate functions (see Figure 1 and Equation (6)). The $K$ gate functions extract a $K$-dimensional implicit nonlinear representation of input features. Therefore, CNF can be viewed as a CRF with its inputs being $K$ homogeneous hidden feature-extractors at each position. Similar to CRF, CNF can also be defined on a general graph structure or an high-order Markov chain. This paper mainly focuses on a linear chain CNF model for sequence labeling.

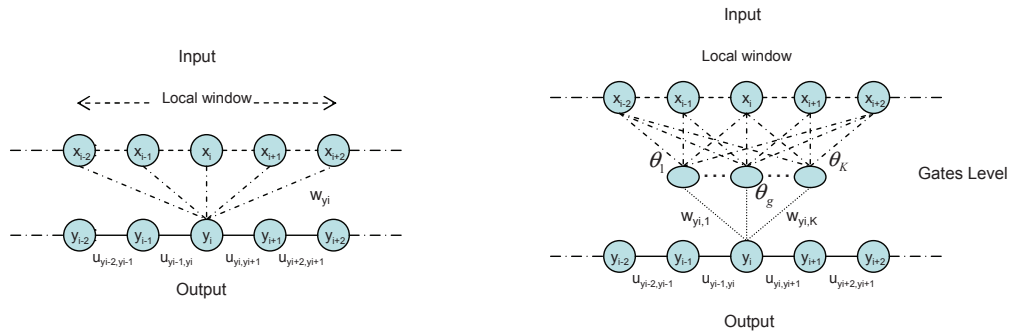

Figure 1: Structures of CRF and CNF

CNF can also be viewed as a natural combination of neural networks and log-linear models. In the hidden layer, there are a set of neurons that extract implicit features from input. Then the log-linear model in the output layer utilizes the implicit features as its input. The parameters in the hidden neurons and the log-linear model can be jointly optimized. After learning the parameters, we can first compute all the hidden neuron values from the input and then use an inference algorithm to predict the output. Any inference algorithm used by CRF, such as Viterbi [7], can be used by CNF. Assume that the dimension of feature vector at each vertex is $D$. The computational complexity for the $K$ neurons is $O(NKD)$. Supposing Viterbi is used as the inference algorithm, the total computational complexity of CNF inference is $O(NMK + NKD)$. Empirically the number of hidden neurons $K$ is small, so the CNF inference procedure may have lower computational complexity than CRF. In our experiments, CNF shows superior predictive performance over two baseline methods: neural networks and CRF.

## 4 Parameter Optimization

Similar to CRF, we can use the maximum likelihood method to train the model parameters such that the log-likelihood is maximized. For CNF, the log-likelihood is as follows.

$$\log P(Y|X) = \sum_{t=1}^{N} (\psi(Y, X, t) + \phi(Y, X, t))) - \log Z(X) \tag{7}$$

Since CNF contains a hidden layer of gate function $h$, the log-likelihood function is not convex any more. Therefore, it is very likely that we can only obtain a local optimal solution of the parameters. Although both the output and hidden layers contain model parameters, all the parameters can be learned together by gradient-based optimization. We can use LBFGS [12] as the optimization routine to search for the optimal model parameters because 1) LBFGS is very efficient and robust; and 2) LBFGS provides us an approximation of inverse Hessian for hyperparameter learning [13], which will be described in the next section. The gradient of the log-likelihood with respect to the parameters is given by

$$\frac{\partial \log P}{\partial u_{y,y'}} = \sum_{t=1}^{N} \delta[y_t = y]\delta[y_{t-1} = y'] - \mathbf{E}_{P(\tilde{Y}|X,w,u,\theta)}[\sum_{t=1}^{N} \delta[\tilde{y}_t = y]\delta[\tilde{y}_{t-1} = y']] \qquad (8)$$

$$\frac{\partial \log P}{\partial w_{y,g}} = \sum_{t=1}^{N} \delta[y_t = y]h(\theta_g^T \mathbf{f}(X,t)) - \mathbf{E}_{P(\tilde{Y}|X,w,u,\theta)}[\sum_{t=1}^{N} \delta[\tilde{y}_t = y]h(\theta_g^T \mathbf{f}(X,t))] \qquad (9)$$

$$\frac{\partial \log P}{\partial \theta_g} = \sum_{t=1}^{N} w_{y_t,g}\frac{\partial h(\theta_g^T \mathbf{f}(X,t))}{\partial \theta_g} - \mathbf{E}_{P(\tilde{Y}|X,w,u,\theta)}[\sum_{t=1}^{N} w_{\tilde{y}_t,g}\frac{\partial h(\theta_g^T \mathbf{f}(X,t))}{\partial \theta_g}] \qquad (10)$$

where $\delta$ is the indicator function.

Just like CRF, we can calculate the expectations in these gradients efficiently using the forward-backward algorithm. Assume that the dimension of feature vector at each vertex is $D$. Since the $K$ gate functions can be computed in advance, the computational complexity of the gradient computation is $O(NKD + NM^2K)$ for a single input-output pair with length $N$. If $K$ is smaller than $D$, it is very possible that the computation of gradient in CNF is faster than in CRF, where the complexity of gradient computation is $O(NM^2D)$. In our experiments, $K$ is usually much smaller than $D$. For example, in protein secondary structure prediction, $K = 30$ and $D = 260$. In handwriting recognition, $K = 40$ and $D = 128$. As a result, although the optimization problem is non-convex, the training time of CNF is acceptable. Our experiments show that the training time of CNF is about 2 or 3 times that of CRF.

## 5 Regularization and Hyperparameter Optimization

Because an hidden layer is added to CNF to introduce more expressive power than CRF, it is crucial to control the model complexity of CNF to avoid overfitting. Similar to CRF, we can enforce regularization on the model parameters to avoid overfitting. We assume that the parameters have a Gaussian prior and constrain the inverse covariance matrix (of Gaussian distribution) by a small number of hyperparameters. To simplify the problem, we divide the model parameter vector $\lambda$ into three different groups $w$, $u$ and $\theta$ (see Figure 1) and assume that the parameters among different groups are independent of each other. Furthermore, we assume parameters in each group share the same Gaussian prior with a diagonal covariance matrix. Let $\alpha = [\alpha_w, \alpha_u, \alpha_\theta]^T$ denote the vector of three regularizations/hyperparameters for these three groups of parameters, respectively. While grid search provides a practical way to determine the best value at low resolution for a single hyperparameter, we need a more sophisticated method to determine three hyperparameters simultaneously. In this section, we discuss the hyperparameter learning in evidence framework.

### 5.1 Laplace's Approximation

The evidence framework [14] assumes that the posterior of $\alpha$ is sharply peaked around the maximum $\alpha_{max}$. Since no prior knowledge of $\alpha$ is known, the prior of each $\alpha_i, i \in \{w, u, \theta\}$, $P(\alpha_i)$ is chosen to be a constant on log-scale or flat. Thus, the value of $\alpha$ maximizing the posterior of $\alpha$ $P(\alpha|Y,X)$ can be found by maximizing

$$P(Y|X,\alpha) = \int_{\lambda} P(Y|X,\lambda)P(\lambda|\alpha)d\lambda \qquad (11)$$

By Laplace's approximation [14], this integral is approximated around the MAP estimation of weights. We have

$$\log P(Y|X,\alpha) = \log P(Y|X,\lambda_{MAP}) + \log P(\lambda_{MAP}|\alpha) - \frac{1}{2}\log \det(A) + \mathbf{const} \qquad (12)$$

where $A$ is the hessian of $\log P(Y|X, \lambda_{MAP}) + \log P(\lambda_{MAP}|\alpha)$ with respect to $\lambda$.

In order to maximize the approximation, we take the derivative of the right hand side of Equation (12) with respect to $\alpha$. The optimal $\alpha$ value can be derived by the following update formula.

$$\alpha_i^{new} = \frac{1}{\lambda_{MAP}^T \lambda_{MAP}}(W_i - \alpha_i^{old}\mathbf{Tr}(A^{-1})) \tag{13}$$

where $W_i$ is the number of parameters in group $i \in \{w, u, \theta\}$.

### 5.2 Approximation of the Trace of Inverse Hessian

When there is a large number of model parameters, accurate computation of $\mathbf{Tr}(A^{-1})$ is very expensive. All model parameters are coupled together by the normalization factor, so the diagonal approximation of Hessian or the outer-product approximation are not appropriate. In this work, we approximate inverse Hessian using information available in the parameter optimization procedure. The LBFGS algorithm is used to optimize parameters iteratively, so we can approximate inverse Hessian at $\lambda_{MAP}$ using the update information generated in the past several iterations. This approach is also employed in [15, 14]. From the LBFGS update formula [13], we can compute the approximation of the trace of inverse Hessian very efficiently. The computational complexity of this approximation is only $O(m^3 + nm^2)$, while the accurate computation has complexity $O(n^3)$ where $n$ is the number of parameters and $m$ is the size of history budget used by LBFGS. Since $m$ is usually much smaller than $n$, the computational complexity is only $O(nm^2)$. See Theorem 2.2 in [13] for more detailed account of this approximation method.

### 5.3 Hyperparameter Update

The hyperparameter $\alpha$ is iteratively updated by a two-step procedure. In the first step we fix hyperparameter $\alpha$ and optimize the model parameters by maximizing the log-likelihood in Equation (7) using LBFGS. In the second step,we fix the model parameters and then update $\alpha$ using Equation (13). This two-step procedure is iteratively carried out until the norm of $\alpha$ does not change more than a threshold. Figure 2 shows the learning curve of the hyperparameter on a protein secondary structure prediction benchmark. In our experiments, the update usually converges in less than 15 iterations. Also we found that this method achieves almost the same test performance as the grid search approach on two public benchmarks.

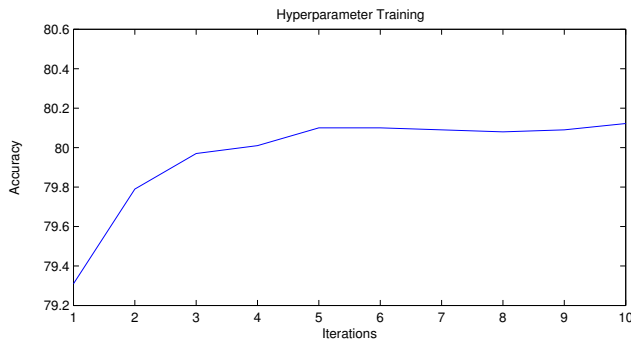

Figure 2: Learning curve of hyperparameter $\alpha$.

## 6 Related Work

Most existing methods for sequence labeling are built under the framework of graphical models such as HMM and CRF. Since these approaches are incapable of capturing highly complex relationship between observations and labels, many structured models are proposed for nonlinear modeling of label-observation dependency. For example, kernelized max margin Markov networks [8], SVM-struct [9] and kernel CRF [16] use nonlinear kernels to model the complex relationship between

observations and labels. Although these kernelized models are convex, it is still too expensive to train and test them in the case that observations are of very high dimension. Furthermore, the number of resultant support vectors for these kernel methods are also very large. Instead, CNF has computational complexity comparable to CRF. Although CNF is non-convex and usually only the local minimum solution can be obtained, CNF still achieves very good performance in real-world applications. Very recently, the probabilistic neural language model [17] and recurrent temporal restricted Boltzmann machine [18] are proposed for natural language and time series modeling. These two methods model sequential data using a directed graph structure, so they are essentially generative models. By contrast, our CNF is a discriminative model, which is mainly used for discriminative prediction of sequence data. The hierarchical recurrent neural networks [19, 20] can be viewed as a hybrid of HMM and neural networks (HMM/NN), building on a directed linear chain. Similarly, CNF can be viewed as an a hybrid of CRF and neural networks, which has the global normalization factor and alleviate the label-bias problem.

## 7 Experiments

### 7.1 Protein Secondary Structure Prediction

Protein secondary structure (SS) prediction is a fundamental problem in computational biology as well as a typical problem used to evaluate sequence labeling methods. Given a protein sequence consisting of a collection of residues, the problem of protein SS prediction is to predict the secondary structure type at each residue. A variety of methods have been described in literature for protein SS prediction.

Given a protein sequence, we first run PSI-BLAST [21] to generate sequence profile and then use this profile as input to predict SS. A sequence profile is a position-specific scoring matrix $X$ with $n \times 20$ elements where $n$ is the number of residues in a protein. Formally, $X = [x_1, x_2, x_3, ..., x_n]$ where $x_i$ is a vector of 20 elements. Each $x_i$ contains 20 position-specific scores, each corresponding to one of the 20 amino acids in nature. The output we want to predict is $Y = [y_1, y_2, ..., y_n]$ where $y_i \in \{H, E, C\}$ represents the secondary structure type at the $i^{th}$ residue.

We evaluate all the SS prediction methods using the CB513 benchmark [22], which consists of 513 no-homologous proteins. The true secondary structure for each protein is calculated using DSSP [23], which generates eight possible secondary structure states. Then we convert these 8 states into three SS types as follows: H and G to H (Helix), B and E to E (Sheets) and all other states to C (Coil). Q3 is used to measure the accuracy of three SS types averaged on all positions. To obtain good performance, we also linearly transform $X$ into values in $[0, 1]$ as suggested by Kim et al[24].

$$S(x) = \begin{cases} 0 & \text{if } x < -5; \\ 0.1x+0.5 & \text{if } -5 \leq x \leq 5; \\ 1 & \text{if } x > 5. \end{cases}$$

To determine the number of gate functions for CNF, we enumerate this number in set $\{10,20,30,40,60,100\}$. We also enumerate window size for CNF in set $\{7,9,11,13,15,17\}$ and find that the best evidence is achieved when window size is 13 and $K = 30$. Two baseline methods are used for comparison: conditional random fields and neural networks. All the parameters of these methods are carefully tuned. The best window sizes for neural networks and CRF are 15 and 13, respectively. We also compared our methods with other popular secondary structure prediction programs. CRF, neural networks, Semi-Markov HMM [25], SVMpsi [24], PSIPRED[2] and CNF use the sequence profile generated by PSI-BLAST as described above. SVMpro [26] uses the position specific frequency as input feature. YASSPP [27] and SPINE [28] also use other residue-specific features in addition to sequence profile.

Table 1 lists the overall performance of a variety of methods on the CB513 data set. As shown in this table, there are two types of gains on accuracy. First, by using one hidden layer to model the nonlinear relationship between input and output, CNF achieves a very significant gain over linear chain CRF. This also confirms that strong nonlinear relationship exists between sequence profile and secondary structure type. Second, by modeling interdependency between neighbor residues, CNF also obtains much better prediction accuracy over neural networks. We also tested the the hybrid of HMM/NN on this dataset. The predicted accuracy of HMM/NN is about three percent less than

Table 1: Performance of various methods for protein secondary structure prediction on the CB513 dataset. Semi-Markov HMM is a segmental semi-Markov model for sequence labeling. SVMpro and SVMpsi are jury method with the SVM (Gaussian kernel) as the basic classifiers. YASSPP use the SVM with a specifically designed profile kernel function for SVM classifiers. PSIPRED is a two stage double-hidden layer neural network. SPINE is voting systems with multiple coupled neural networks. YASSPP, PSIPRED and SPINE also use other features besides the PSSM scores. An * symbol indicates the methods are tested over a 10-fold cross-validation on CB513, while others are tested over a 7-fold cross-validation.

| Methods | Q3(%) |
|---|---|
| Conditional Random Fields | 72.9 |
| SVM-struct (Linear Kernel) | 73.1 |
| Neural Networks (one hidden layer) | 72 |
| Neural Networks (two hidden layer) | 74 |
| Semimarkov HMM | 72.8 |
| SVMpro | 73.5 |
| SVMpsi | 76.6 |
| PSIPRED | 76 |
| YASSPP | 77.8 |
| SPINE* | 76.8 |
| Conditional Neural Fields | **80.1** $\pm0.3$ |
| Conditional Neural Fields* | **80.5** $\pm0.3$ |

that of CNF. By seamlessly integrating neural networks and CRF, CNF outperforms all other the-state-of-art prediction methods on this dataset. We also tried Max-Margin Markov Network [8] and SVM-struct[1] with RBF kernel for this dataset. However, because the dataset is large and the feature space is of high dimension, it is impossible for these kernel-based methods to finish training within a reasonable amount of time. Both of them failed to converge within 120 hours. The running time of CNF learning and inference is about twice that of CRF.

## 7.2 Handwriting Recognition

Handwriting recognition(OCR) is another widely-used benchmark for sequence labeling algorithms. We use the subset of OCR dataset chosen by Taskar [8], which contains 6876 sequences. In this dataset, each word consists of a sequence of characters and each character is represented by an image with $16 \times 8$ binary pixels. In addition to using the vector of pixel values as input features, we do not use any higher-level features. Formally, the input $X = [x_1, x_2, x_3, ..., x_n]$ is a sequence of 128-dimensional binary vectors. The output we want to predict is a sequence of labels. Each label $y_i$ for image $x_i$ is one of the 26 classes $\{a, b, c, ..., z\}$. The accuracy is defined as the average accuracy over all characters.

The number of gate functions for CNF is selected from set $\{10, 20, 30, 40, 60, 100\}$ and we find that the best evidence is achieved when $K = 40$. Window sizes for all methods are fixed to 1. All the methods are tested using 10-fold cross-validation and their performance are shown in Table 2. As shown in this table, CNF achieves superior performance over log-linear methods, SVM, CRF and neural networks. CNF is also comparable with two slightly different max margin Markov network models.

## 8   Discussion

We present a probabilistic graphical model conditional neural fields (CNF) for sequence labeling tasks which require accurate account of nonlinear relationship between input and output. CNF is a very natural integration of conditional graphical models and neural networks and thus, inherits advantages from both of them. On one hand, by neural networks, CNF can model nonlinear relationship between input and output. On the other hand, by using graphical representation, CNF

Table 2: Performance of various methods on handwriting recognition. The results of logistic regression, SVM and max margin Markov networks are taken from [8]. Both CNF and neural networks use 40 neurons in the hidden layer. The CRF performance (78.9%) we obtained is a bit better than 76% in [8].

| Methods | Accuracy(%) |
|---|---|
| Logistic Regression | 71 |
| SVM (linear) | 71 |
| SVM (quadratic) | 80 |
| SVM (cubic) | 81 |
| SVM-struct | 80 |
| Conditional Random Fields | 78.9 |
| Neural Networks | 79.8 |
| MMMN (linear) | 80 |
| MMMN (quadratic) | 87 |
| MMMN (cubic) | 87 |
| Conditional Neural Fields | 86.9 $\pm$0.4 |

can model interdependency between output labels. While CNF is more sophisticated and expressive than CRF, the computational complexity of learning and inference is not necessarily higher. Our experimental results on large-scale datasets indicate that CNF can be trained and tested as almost efficient as CRF but much faster than kernel-based methods. Although CNF is not convex, it can still be trained using the quasi-Newton method to obtain a local optimal solution, which usually works very well in real-world applications.

In two real-world applications, CNF significantly outperforms two baseline methods, CRF and neural networks. On protein secondary structure prediction, CNF achieves the best performance over all methods we tested. on handwriting recognition, CNF also compares favorably with the best method max-margin Markov network. We are currently generalizing our CNF model to a second-order Markov chain and a more general graph structure and also studying if it will improve predictive power of CNF by interposing more than one hidden layers between input and output.

**Acknowledgements**

We thank Nathan Srebro and David McAllester for insightful discussions.

## Footnotes

[1]http://svmlight.joachims.org/svm_struct.html

# References

[1] Fei Sha and O. Pereira. Shallow parsing with conditional random fields. In *Proceedings of Human Language Technology-NAACL 2003*.

[2] D. T. Jones. Protein secondary structure prediction based on position-specific scoring matrices. *Journal of Molecular Biology*, 292(2):195–202, September 1999.

[3] Feng Zhao, Shuaicheng Li, Beckett W. Sterner, and Jinbo Xu. Discriminative learning for protein conformation sampling. *Proteins*, 73(1):228–240, October 2008.

[4] Feng Zhao, Jian Peng, Joe Debartolo, Karl F. Freed, Tobin R. Sosnick, and Jinbo Xu. A probabilistic graphical model for ab initio folding. In *RECOMB 2'09: Proceedings of the 13th Annual International Conference on Research in Computational Molecular Biology*, pages 59–73, Berlin, Heidelberg, 2009. Springer-Verlag.

[5] Sy Bor Wang, Ariadna Quattoni, Louis-Philippe Morency, and David Demirdjian. Hidden conditional random fields for gesture recognition. In *CVPR 2006*.

[6] Lawrence R. Rabiner. A tutorial on hidden markov models and selected applications in speech recognition. In *Proceedings of the IEEE*, 1989.

[7] John D. Lafferty, Andrew McCallum, and Fernando C. N. Pereira. Conditional random fields: Probabilistic models for segmenting and labeling sequence data. In *ICML 2001*.

[8] Ben Taskar, Carlos Guestrin, and Daphne Koller. Max-margin markov networks. In *NIPS 2003*.

[9] Ioannis Tsochantaridis, Thomas Hofmann, Thorsten Joachims, and Yasemin Altun. Support vector machine learning for interdependent and structured output spaces. In *ICML 2004*.

[10] Nam Nguyen and Yunsong Guo. Comparisons of sequence labeling algorithms and extensions. In *ICML 2007*.

[11] Yan Liu, Jaime Carbonell, Judith Klein-Seetharaman, and Vanathi Gopalakrishnan. Comparison of probabilistic combination methods for protein secondary structure prediction. *Bioinformatics*, 20(17), November 2004.

[12] D. C. Liu and J. Nocedal. On the limited memory bfgs method for large scale optimization. *Mathematical Programming*, 45(3), 1989.

[13] Richard H. Byrd, Jorge Nocedal, and Robert B. Schnabel. Representations of quasi-newton matrices and their use in limited memory methods. *Mathematical Programming*, 63(2), 1994.

[14] David J. C. Mackay. A practical bayesian framework for backpropagation networks. *Neural Computation*, 4:448–472, 1992.

[15] Christopher M. Bishop. *Neural Networks for Pattern Recognition*. Oxford University Press, November 1995.

[16] John Lafferty, Xiaojin Zhu, and Yan Liu. Kernel conditional random fields: representation and clique selection. In *ICML 2004*.

[17] Yoshua Bengio, Réjean Ducharme, Pascal Vincent, and Christian Janvin. A neural probabilistic language model. *Journal of Machine Learning Research*, 3:1137–1155, 2003.

[18] Ilya Sutskever, Geoffrey E Hinton, and Graham Taylor. The recurrent temporal restricted boltzmann machine. In D. Koller, D. Schuurmans, Y. Bengio, and L. Bottou, editors, *NIPS 2009*.

[19] Barbara Hammer. Recurrent networks for structured data - a unifying approach and its properties. *Cognitive Systems Research*, 2002.

[20] Alex Graves and Juergen Schmidhuber. Offline handwriting recognition with multidimensional recurrent neural networks. In D. Koller, D. Schuurmans, Y. Bengio, and L. Bottou, editors, *NIPS 2009*.

[21] S. F. Altschul, T. L. Madden, A. A. Schäffer, J. Zhang, Z. Zhang, W. Miller, and D. J. Lipman. Gapped blast and psi-blast: a new generation of protein database search programs. *Nucleic Acids Research*, 25, September 1997.

[22] James A. Cuff and Geoffrey J. Barton. Evaluation and improvement of multiple sequence methods for protein secondary structure prediction. *Proteins: Structure, Function, and Genetics*, 34, 1999.

[23] Wolfgang Kabsch and Christian Sander. Dictionary of protein secondary structure: Pattern recognition of hydrogen-bonded and geometrical features. *Biopolymers*, 22(12):2577–2637, December 1983.

[24] H. Kim and H. Park. Protein secondary structure prediction based on an improved support vector machines approach. *Protein Engineering*, 16(8), August 2003.

[25] Wei Chu, Zoubin Ghahramani, and David. A graphical model for protein secondary structure prediction. In *ICML 2004*.

[26] Sujun Hua and Zhirong Sun. A novel method of protein secondary structure prediction with high segment overlap measure: Support vector machine approach. *Journal of Molecular Biology*, 308, 2001.

[27] George Karypis. Yasspp: Better kernels and coding schemes lead to improvements in protein secondary structure prediction. *Proteins: Structure, Function, and Bioinformatics*, 64(3):575–586, 2006.

[28] O. Dor and Y. Zhou. Achieving 80% ten-fold cross-validated accuracy for secondary structure prediction by large-scale training. *Proteins: Structure, Function, and Bioinformatics*, 66, March 2007.

